# The Kernel Beta Process

**Lu Ren**[*]
Electrical & Computer Engineering Dept.
Duke University
Durham, NC 27708
lr22@duke.edu

**Yingjian Wang**[*]
Electrical & Computer Engineering Dept.
Duke University
Durham, NC 27708
yw65@duke.edu

**David Dunson**
Department of Statistical Science
Duke University
Durham, NC 27708
dunson@stat.duke.edu

**Lawrence Carin**
Electrical & Computer Engineering Dept.
Duke University
Durham, NC 27708
lcarin@duke.edu

## Abstract

A new Lévy process prior is proposed for an uncountable collection of covariate-dependent feature-learning measures; the model is called the kernel beta process (KBP). Available covariates are handled efficiently via the kernel construction, with covariates assumed observed with each data sample ("customer"), and latent covariates learned for each feature ("dish"). Each customer selects dishes from an infinite buffet, in a manner analogous to the beta process, with the added constraint that a customer first decides probabilistically whether to "consider" a dish, based on the distance in covariate space between the customer and dish. If a customer *does* consider a particular dish, that dish is then selected probabilistically as in the beta process. The beta process is recovered as a limiting case of the KBP. An efficient Gibbs sampler is developed for computations, and state-of-the-art results are presented for image processing and music analysis tasks.

## 1 Introduction

Feature learning is an important problem in statistics and machine learning, characterized by the goal of (typically) inferring a low-dimensional set of features for representation of high-dimensional data. It is desirable to perform such analysis in a nonparametric manner, such that the number of features may be learned, rather than *a priori* set. A powerful tool for such learning is the Indian buffet process (IBP) [4], in which the data samples serve as "customers", and the potential features serve as "dishes". It has recently been demonstrated that the IBP corresponds to a marginalization of a beta-Bernoulli process [15]. The IBP and beta-Bernoulli constructions have found significant utility in factor analysis [7, 17], in which one wishes to infer the number of factors needed to represent data of interest. The beta process was developed originally by Hjort [5] as a Lévy process prior for "hazard measures", and was recently extended for use in feature learning [15], the interest of this paper; we therefore here refer to it as a "feature-learning measure."

The beta process is an example of a Lévy process [6], another example of which is the gamma process [1]; the *normalized* gamma process is well known as the Dirichlet process [3, 14]. A key characteristic of such models is that the data samples are assumed exchangeable, meaning that the order/indices of the data may be permuted with no change in the model.

---

[*]The first two authors contributed equally to this work.

An important line of research concerns removal of the assumption of exchangeability, allowing incorporation of covariates (*e.g.*, spatial/temporal coordinates that may be available with the data). As an example, MacEachern introduced the *dependent* Dirichlet process [8]. In the context of feature learning, the phylogenetic IBP removes the assumption of sample exchangeability by imposing prior knowledge on inter-sample relationships via a tree structure [9]. The form of the tree may be constituted as a result of covariates that are available with the samples, but the tree is not necessarily unique. A dependent IBP (dIBP) model has been introduced recently, with a hierarchical Gaussian process (GP) used to account for covariate dependence [16]; however, the use of a GP may constitute challenges for large-scale problems. Recently a dependent hierarchical beta process (dHBP) has been developed, yielding encouraging results [18]. However, the dHBP has the disadvantage of assigning a kernel to each data sample, and therefore it scales unfavorably as the number of samples increases.

In this paper we develop a new Lévy process prior, termed the kernel beta process (KBP), which yields an uncountable number of *covariate-dependent* feature-learning measures, with the beta process a special case. This model may be interpreted as inferring covariates $\boldsymbol{x}_i^*$ for each feature (dish), indexed by $i$. The generative process by which the $n$th data sample, with covariates $\boldsymbol{x}_n$, selects features may be viewed as a two-step process. First the $n$th customer (data sample) decides whether to "examine" dish $i$ by drawing $z_{ni}^{(1)} \sim \text{Bernoulli}(K(\boldsymbol{x}_n, \boldsymbol{x}_i^*; \psi_i^*))$, where $\psi_i^*$ are dish-dependent kernel parameters that are also inferred (the $\{\psi_i^*\}$ defining the meaning of proximity/locality in covariate space). The kernels are designed to satisfy $K(\boldsymbol{x}_n, \boldsymbol{x}_i^*; \psi_i^*) \in (0,1]$, $K(\boldsymbol{x}_i^*, \boldsymbol{x}_i^*; \psi_i^*) = 1$, and $K(\boldsymbol{x}_n, \boldsymbol{x}_i^*; \psi_i^*) \to 0$ as $\|\boldsymbol{x}_n - \boldsymbol{x}_i^*\|_2 \to \infty$. In the second step, if $z_{ni}^{(1)} = 1$, customer $n$ draws $z_{ni}^{(2)} \sim \text{Bernoulli}(\pi_i)$, and if $z_{ni}^{(2)} = 1$, the feature associated with dish $i$ is employed by data sample $n$. The parameters $\{\boldsymbol{x}_i^*, \psi_i^*, \pi_i\}$ are inferred by the model. After computing the posterior distribution on model parameters, the number of kernels required to represent the measures is defined by the number of features employed from the buffet (typically small relative to the data size); this is a significant computational savings relative to [18, 16], for which the complexity of the model is tied to the number of data samples, even if a small number of features are ultimately employed.

In addition to introducing this new Lévy process, we examine its properties, and demonstrate how it may be efficiently applied in important data analysis problems. The hierarchical construction of the KBP is fully conjugate, admitting convenient Gibbs-sampling (complicated sampling methods were required for the method in [18]). To demonstrate the utility of the model we consider image-processing and music-analysis applications, for which state-of-the-art performance is demonstrated compared to other relevant methods.

## 2 Kernel Beta Process

### 2.1 Review of beta and Bernoulli processes

A beta process $B \sim \text{BP}(c, B_0)$ is a distribution on positive random measures over the space $(\Omega, \mathcal{F})$. Parameter $c(\omega)$ is a positive function over $\omega \in \Omega$, and $B_0$ is the *base measure* defined over $\Omega$. The beta process is an example of a Lévy process, and the Lévy measure of $\text{BP}(c, B_0)$ is

$$\nu(d\pi, d\omega) = c(\omega)\pi^{-1}(1-\pi)^{c(\omega)-1}d\pi B_0(d\omega) \tag{1}$$

To draw $B$, one draws a set of points $(\omega_i, \pi_i) \in \Omega \times [0,1]$ from a Poisson process with measure $\nu$, yielding

$$B = \sum_{i=1}^{\infty} \pi_i \delta_{\omega_i} \tag{2}$$

where $\delta_{\omega_i}$ is a unit point measure at $\omega_i$; $B$ is therefore a discrete measure, with probability one. The infinite sum in (2) is a consequence of drawing Poisson($\lambda$) atoms $\{\omega_i, \pi_i\}$, with $\lambda = \int_\Omega \int_{[0,1]} \nu(d\omega, d\pi) = \infty$. Additionally, for any set $\mathcal{A} \subset \mathcal{F}$, $B(\mathcal{A}) = \sum_{i: \omega_i \in \mathcal{A}} \pi_i$.

If $Z_n \sim \text{BeP}(B)$ is the $n$th draw from a Bernoulli process, with $B$ defined as in (2), then

$$Z_n = \sum_{i=1}^{\infty} b_{ni}\delta_{\omega_i} , \quad b_{ni} \sim \text{Bernoulli}(\pi_i) \tag{3}$$

A set of $N$ such draws, $\{Z_n\}_{n=1,N}$, may be used to define whether feature $\omega_i \in \Omega$ is utilized to represent the $n$th data sample, where $b_{ni} = 1$ if feature $\omega_i$ is employed, and $b_{ni} = 0$ otherwise. One may marginalize out the measure $B$ analytically, yielding conditional probabilities for the $\{Z_n\}$ that correspond to the Indian buffet process [15, 4].

## 2.2 Covariate-dependent Lévy process

In the above beta-Bernoulli construction, the same measure $B \sim \mathrm{BP}(c, B_0)$ is employed for generation of all $\{Z_n\}$, implying that each of the $N$ samples have the same probabilities $\{\pi_i\}$ for use of the respective features $\{\omega_i\}$. We now assume that with each of the $N$ samples of interest there are an associated set of covariates, denoted respectively as $\{\boldsymbol{x}_n\}$, with each $\boldsymbol{x}_n \in \mathcal{X}$. We wish to impose that if samples $n$ and $n'$ have similar covariates $\boldsymbol{x}_n$ and $\boldsymbol{x}_{n'}$, that it is probable that they will employ a similar subset of the features $\{\omega_i\}$; if the covariates are distinct it is less probable that feature sharing will be manifested.

Generalizing (2), consider

$$\mathcal{B} = \sum_{i=1}^{\infty} \gamma_i \delta_{\omega_i} , \quad \omega_i \sim B_0 \tag{4}$$

where $\gamma_i = \{\gamma_i(\boldsymbol{x}) : \boldsymbol{x} \in \mathcal{X}\}$ is a stochastic process (random function) from $\mathcal{X} \rightarrow [0, 1]$ (drawn independently from the $\{\omega_i\}$). Hence, $\mathcal{B}$ is a dependent *collection* of Lévy processes with the measure specific to covariate $\boldsymbol{x} \in \mathcal{X}$ being $\mathcal{B}_{\boldsymbol{x}} = \sum_{i=1}^{\infty} \gamma_i(\boldsymbol{x})\delta_{\omega_i}$. This constitutes a general specification, with several interesting special cases. For example, one might consider $\gamma_i(\boldsymbol{x}) = g\{\mu_i(\boldsymbol{x})\}$, where $g : \mathbb{R} \rightarrow [0, 1]$ is any monotone differentiable link function and $\mu_i(\boldsymbol{x}) : \mathcal{X} \rightarrow \mathbb{R}$ may be modeled as a Gaussian process [10], or related kernel-based construction. To choose $g\{\mu_i(\boldsymbol{x})\}$ one can potentially use models for the predictor-dependent breaks in probit, logistic or kernel stick-breaking processes [13, 11, 2]. In the remainder of this paper we propose a special case for design of $\gamma_i(\boldsymbol{x})$, termed the *kernel* beta process (KBP).

## 2.3 Characteristic function of the kernel beta process

Recall from Hjort [5] that $B \sim \mathrm{BP}(c(\omega), B_0)$ is a beta process on measure space $(\Omega, \mathcal{F})$ if its characteristic function satisfies

$$\mathbb{E}[e^{juB(\mathcal{A})}] = \exp\{\int_{[0,1]\times\mathcal{A}} (e^{ju\pi} - 1)\nu(d\pi, d\omega)\} \tag{5}$$

where here $j = \sqrt{-1}$, and $\mathcal{A}$ is any subset in $\mathcal{F}$. The beta process is a particular class of the Lévy process, with $\nu(d\pi, d\omega)$ defined as in (1).

For kernel $K(\boldsymbol{x}, \boldsymbol{x}^*; \psi^*)$, let $\boldsymbol{x} \in \mathcal{X}$, $\boldsymbol{x}^* \in \mathcal{X}$, and $\psi^* \in \Psi$; it is assumed that $K(\boldsymbol{x}, \boldsymbol{x}^*; \psi^*) \in [0, 1]$ for all $\boldsymbol{x}$, $\boldsymbol{x}^*$ and $\psi^*$. As a specific example, for the radial basis function $K(\boldsymbol{x}, \boldsymbol{x}^*; \psi^*) = \exp[-\psi^* \|\boldsymbol{x} - \boldsymbol{x}^*\|_2]$, where $\psi^* \in \mathbb{R}^+$. Let $\boldsymbol{x}^*$ represent random variables drawn from probability measure $H$, with support on $\mathcal{X}$, and $\psi^*$ is also a random variable drawn from an appropriate probability measure $Q$ with support over $\Psi$ (*e.g.*, in the context of the radial basis function, $\psi^*$ are drawn from a probability measure with support over $\mathbb{R}^+$). We now define a new Lévy measure

$$\nu_{\mathcal{X}} = H(d\boldsymbol{x}^*)Q(d\psi^*)\nu(d\pi, d\omega) \tag{6}$$

where $\nu(d\pi, d\omega)$ is the Lévy measure associated with the beta process, defined in (1).

**Theorem 1** Assume parameters $\{\boldsymbol{x}_i^*, \psi_i^*, \pi_i, \omega_i\}$ are drawn from measure $\nu_{\mathcal{X}}$ in (6), and that the following measure is constituted

$$\mathcal{B}_{\boldsymbol{x}} = \sum_{i=1}^{\infty} \pi_i K(\boldsymbol{x}, \boldsymbol{x}_i^*; \psi_i^*)\delta_{\omega_i} \tag{7}$$

which may be evaluated for *any* covariate $\boldsymbol{x} \in \mathcal{X}$. For any finite set of covariates $\mathcal{S} = \{\boldsymbol{x}_1, \ldots, \boldsymbol{x}_{|\mathcal{S}|}\}$, we define the $|\mathcal{S}|$-dimensional random vector $\boldsymbol{K} = (K(\boldsymbol{x}_1, \boldsymbol{x}^*; \psi^*), \ldots, K(\boldsymbol{x}_{|\mathcal{S}|}, \boldsymbol{x}^*; \psi^*))^T$, with random variables $\boldsymbol{x}^*$ and $\psi^*$ drawn from $H$ and $Q$, respectively. For any set $\mathcal{A} \subset \mathcal{F}$, the $\mathcal{B}$ evaluated at covariates $\mathcal{S}$, on the set $\mathcal{A}$,

yields an $|\mathcal{S}|$-dimensional random vector $\boldsymbol{\mathcal{B}}(\mathcal{A}) = (\mathcal{B}_{\boldsymbol{x}_1}(\mathcal{A}), \ldots, \mathcal{B}_{\boldsymbol{x}_{|\mathcal{S}|}}(\mathcal{A}))^T$, where $\mathcal{B}_{\boldsymbol{x}}(\mathcal{A}) = \sum_{i:\ \omega_i \in \mathcal{A}} \pi_i K(\boldsymbol{x}, \boldsymbol{x}_i^*; \psi_i^*)$. Expression (7) is a covariate-dependent Lévy process with Lévy measure (6), and characteristic function for an arbitrary set of covariates $\mathcal{S}$ satisfying

$$\mathbb{E}[e^{j<\boldsymbol{u}, \boldsymbol{\mathcal{B}}(\mathcal{A})>}] = \exp\{\int_{\mathcal{X} \times \Psi \times [0,1] \times \mathcal{A}} (e^{j<\boldsymbol{u}, \boldsymbol{K}\pi>} - 1)\nu_{\mathcal{X}}(d\boldsymbol{x}^*, d\psi^*, d\pi, d\omega)\} \tag{8}$$

□

A proof is provided in the Supplemental Material. Additionally, for notational convenience, below a draw of (7), valid for all covariates in $\mathcal{X}$, is denoted $\mathcal{B} \sim \text{KBP}(c, B_0, H, Q)$, with $c$ and $B_0$ defining $\nu(d\pi, d\omega)$ in (1).

### 2.4 Relationship to the beta-Bernoulli process

If the covariate-dependent measure $\mathcal{B}_{\boldsymbol{x}}$ in (7) is employed to define covariate-dependent feature usage, then $Z_{\boldsymbol{x}} \sim \text{BeP}(\mathcal{B}_{\boldsymbol{x}})$, generalizing (3). Hence, given $\{\boldsymbol{x}_i^*, \psi_i^*, \pi_i\}$, the feature-usage measure is $Z_{\boldsymbol{x}} = \sum_{i=1}^{\infty} b_{\boldsymbol{x}i}\delta_{\omega_i}$, with $b_{\boldsymbol{x}i} \sim \text{Bernoulli}(\pi_i K(\boldsymbol{x}, \boldsymbol{x}_i^*; \psi_i^*))$. Note that it is equivalent in distribution to express $b_{\boldsymbol{x}i} = z_{\boldsymbol{x}i}^{(1)} z_{\boldsymbol{x}i}^{(2)}$, with $z_{\boldsymbol{x}i}^{(1)} \sim \text{Bernoulli}(K(\boldsymbol{x}, \boldsymbol{x}_i^*; \psi_i^*))$ and $z_{\boldsymbol{x}i}^{(2)} \sim \text{Bernoulli}(\pi_i)$. This model therefore yields the two-step generalization of the generative process of the beta-Bernoulli process discussed in the Introduction. The condition $z_{\boldsymbol{x}i}^{(1)} = 1$ only has a high probability when observed covariates $\boldsymbol{x}$ are near the (latent/inferred) covariates $\boldsymbol{x}_i^*$. It is deemed attractive that this intuitive generative process comes as a result of a rigorous Lévy process construction, the properties of which are summarized next.

### 2.5 Properties of $\mathcal{B}$

For all Borel subsets $\mathcal{A} \in \mathcal{F}$, if $\mathcal{B}$ is drawn from the KBP and for covariates $\boldsymbol{x}, \boldsymbol{x}' \in \mathcal{X}$, we have

$$\mathbb{E}[\mathcal{B}_{\boldsymbol{x}}(\mathcal{A})] = B_0(\mathcal{A})\mathbb{E}(K_{\boldsymbol{x}})$$

$$\text{Cov}(\mathcal{B}_{\boldsymbol{x}}(\mathcal{A}), \mathcal{B}_{\boldsymbol{x}'}(\mathcal{A})) = \mathbb{E}(K_{\boldsymbol{x}}K_{\boldsymbol{x}'}) \int_{\mathcal{A}} \frac{B_0(d\omega)(1 - B_0(d\omega))}{c(\omega) + 1} - \text{Cov}(K_{\boldsymbol{x}}, K_{\boldsymbol{x}'}) \int_{\mathcal{A}} B_0^2(d\omega)$$

where, $\mathbb{E}(K_{\boldsymbol{x}}) = \int_{\mathcal{X} \times \Psi} K(\boldsymbol{x}, \boldsymbol{x}^*; \psi^*)H(dx^*)Q(d\psi^*)$. If $K(\boldsymbol{x}, \boldsymbol{x}^*; \psi^*) = 1$ for all $\boldsymbol{x} \in \mathcal{X}$, $\mathbb{E}(K_{\boldsymbol{x}}) = \mathbb{E}(K_{\boldsymbol{x}}K_{\boldsymbol{x}'}) = 1$, and $\text{Cov}(K_{\boldsymbol{x}}, K_{\boldsymbol{x}'}) = 0$, and the above results reduce to the those for the original BP [15].

Assume $c(\omega) = c$, where $c \in \mathbb{R}^+$ is a constant, and let $\boldsymbol{K}_{\boldsymbol{x}} = (K(\boldsymbol{x}, \boldsymbol{x}_1^*; \psi_1^*), K(\boldsymbol{x}, \boldsymbol{x}_2^*; \psi_2^*), \ldots)^T$ represent an infinite-dimensional vector, then for fixed kernel parameters $\{\boldsymbol{x}_i^*, \psi_i^*\}$,

$$\text{Corr}(\mathcal{B}_{\boldsymbol{x}}(\mathcal{A}), \mathcal{B}_{\boldsymbol{x}'}(\mathcal{A})) = \frac{<\boldsymbol{K}_{\boldsymbol{x}}, \boldsymbol{K}_{\boldsymbol{x}'}>}{\|\boldsymbol{K}_{\boldsymbol{x}}\|_2 \cdot \|\boldsymbol{K}_{\boldsymbol{x}'}\|_2} \tag{9}$$

where it is assumed $<\boldsymbol{K}_{\boldsymbol{x}}, \boldsymbol{K}_{\boldsymbol{x}'}>$, $\|\boldsymbol{K}_{\boldsymbol{x}}\|_2$, $\|\boldsymbol{K}_{\boldsymbol{x}'}\|_2$ are finite; the latter condition is always met when we (in practice) truncate the number of terms used in (7). The expression in (9) clearly imposes the desired property of high correlation in $\mathcal{B}_{\boldsymbol{x}}$ and $\mathcal{B}_{\boldsymbol{x}'}$ when $\boldsymbol{x}$ and $\boldsymbol{x}'$ are proximate.

Proofs of the above properties are provided in the Supplemental Material.

## 3 Applications

### 3.1 Model construction

We develop a covariate-dependent factor model, generalizing [7, 17], which did not consider covariates. Consider data $\boldsymbol{y}_n \in \mathbb{R}^M$ with associated covariates $\boldsymbol{x}_n \in \mathbb{R}^L$, with $n = 1, \ldots, N$. The factor loadings in the factor model here play the role of "dishes" in the buffet analogy, and we model the data as

$$\boldsymbol{y}_n = \mathbf{D}(\boldsymbol{w}_n \circ \boldsymbol{b}_n) + \boldsymbol{\epsilon}_n$$

$$Z_{\boldsymbol{x}_n} \sim \text{BeP}(\mathcal{B}_{\boldsymbol{x}_n}), \quad \mathcal{B} \sim \text{KBP}(c, B_0, H, Q), \quad B_0 \sim \text{DP}(\alpha_0 G_0) \tag{10}$$

$$\boldsymbol{w}_n \sim \mathcal{N}(\mathbf{0}, \alpha_1^{-1}\mathbf{I}_T), \quad \boldsymbol{\epsilon}_n \sim \mathcal{N}(\mathbf{0}, \alpha_2^{-1}\mathbf{I}_M)$$

with gamma priors placed on $\alpha_0$, $\alpha_1$ and $\alpha_2$, with $\circ$ representing the pointwise (Hadamard) vector product, and with $\mathbf{I}_M$ representing the $M \times M$ identity matrix. The Dirichlet process [3] base measure $G_0 = \mathcal{N}(0, \frac{1}{M}\mathbf{I}_M)$, and the KBP base measure $B_0$ is a *mixture* of atoms (factor loadings). For the applications considered it is important that the same atoms be reused at different points $\{\boldsymbol{x}_i^*\}$ in covariate space, to allow for repeated structure to be manifested as a function of space or time, within the image and music applications, respectively. The columns of $\mathbf{D}$ are defined respectively by $(\omega_1, \omega_2, \dots)$ in $\mathcal{B}$, and the vector $\boldsymbol{b}_n = (b_{n1}, b_{n2}, \dots)$ with $b_{nk} = Z_{\boldsymbol{x}_n}(\omega_k)$. Note that $\mathcal{B}$ is drawn *once* from the KBP, and when drawing the $Z_{\boldsymbol{x}_n}$ we evaluate $\mathcal{B}$ as defined by the respective covariate $\boldsymbol{x}_n$.

When implementing the KBP, we truncate the sum in (7) to $T$ terms, and draw the $\pi_i \sim$ Beta$(1/T, 1)$, which corresponds to setting $c = 1$. We set $T$ large, and the model infers the subset of $\{\pi_i\}_{i=1,T}$ that have significant amplitude, thereby estimating the number of factors needed for representation of the data. In practice we let $H$ and $Q$ be multinomial distributions over a discrete and finite set of, respectively, locations for $\{\boldsymbol{x}_i^*\}$ and kernel parameters for $\{\psi_i^*\}$, details of which are discussed in the specific examples.

In (10), the $i$th column of $\mathbf{D}$, denoted $\mathbf{D}_i$, is drawn from $B_0$, with $B_0$ drawn from a Dirichlet process (DP). There are multiple ways to perform such DP clustering, and here we apply the *Pólya urn scheme* [3]. Assume $\mathbf{D}_1, \mathbf{D}_2, \dots, \mathbf{D}_{i-1}$ are a series of i.i.d. random draws from $B_0$, then the successive conditional distribution of $\mathbf{D}_i$ is of the following form:

$$\mathbf{D}_i | \mathbf{D}_1, \dots, \mathbf{D}_{i-1}, \alpha_0, G_0 \sim \sum_{l=1}^{N_u} \frac{n_l^*}{i - 1 + \alpha_0} \delta_{\mathbf{D}_l^*} + \frac{\alpha_0}{i - 1 + \alpha_0} G_0, \quad (11)$$

where $\{\mathbf{D}_l^*\}_{l=1,N_u}$ are the unique dictionary elements shared by the first $i - 1$ columns of $\mathbf{D}$, and $n_l^* = \sum_{j=1}^{i-1} \delta(\mathbf{D}_j = \mathbf{D}_l^*)$. For model inference, an indicator variable $c_i$ is introduced for each $\mathbf{D}_i$, and $c_i = l$ with a probability proportional to $n_l^*$, with $l = 1, \dots, N_u$, with $c_i$ equal to $N_u + 1$ with a probability controlled by $\alpha_0$. If $c_i = l$ for $l = 1, \dots, N_u$, $\mathbf{D}_i$ takes the value $\mathbf{D}_l^*$; otherwise $\mathbf{D}_i$ is drawn from the prior $G_0 = \mathcal{N}(0, \frac{1}{M}\mathbf{I}_M)$, and a new dish/factor loading $\mathbf{D}_{N_u+1}^*$ is hence introduced.

### 3.2 Extensions

It is relatively straightforward to include additional model sophistication into (10), one example of which we will consider in the context of the image-processing example. Specifically, in many applications it is inappropriate to assume a Gaussian model for the noise or residual $\boldsymbol{\epsilon}_n$. In Section 4.3 we consider the following augmented noise model:

$$\boldsymbol{\epsilon}_n = \boldsymbol{\lambda}_n \circ \boldsymbol{m}_n + \hat{\boldsymbol{\epsilon}}_n \quad (12)$$
$$\boldsymbol{\lambda}_n \sim \mathcal{N}(\mathbf{0}, \alpha_\lambda^{-1}\mathbf{I}_M), \quad m_{np} \sim \text{Bernoulli}(\tilde{\pi}_n), \quad \tilde{\pi}_n \sim \text{Beta}(a_0, b_0), \quad \hat{\boldsymbol{\epsilon}}_n \sim \mathcal{N}(\mathbf{0}, \alpha_3^{-1}\mathbf{I}_M)$$

with gamma priors placed on $\alpha_\lambda$ and $\alpha_2$, and with $p = 1, \dots, M$. The term $\boldsymbol{\lambda}_n \circ \boldsymbol{m}_n$ accounts for "spiky" noise, with potentially large amplitude, and $\hat{\pi}_n$ represents the probability of spiky noise in data sample $n$. This type of noise model was considered in [18], with which we compare.

### 3.3 Inference

The model inference is performed with a Gibbs sampler. Due to the limited space, only those variables having update equations distinct from those in the BP-FA of [17] are included here. Assume $T$ is the truncation level for the number of dictionary elements, $\{\mathbf{D}_i\}_{i=1,T}$; $N_u$ is the number of unique dictionary elements values in the current Gibbs iteration, $\{\mathbf{D}_l^*\}_{l=1,N_u}$. For the applications considered in this paper, $K(\boldsymbol{x}_n, \boldsymbol{x}_i^*; \psi_i^*)$ is defined based on the Euclidean distance: $K(\boldsymbol{x}_n, \boldsymbol{x}_i^*; \psi_i^*) = \exp[-\psi_i^* ||\boldsymbol{x}_n - \boldsymbol{x}_i^*||_2]$ for $i = 1, \dots, T$; both $\psi_i^*$ and $\boldsymbol{x}_i^*$ are updated from multinomial distributions (defining $Q$ and $H$, respectively) over a set of discretized values with a uniform prior for each; more details on this are discussed in Sec. 4.

- **Update** $\{\mathbf{D}_l^*\}_{l=1,L}$: $\mathbf{D}_l^* \sim \mathcal{N}(\boldsymbol{\mu}_l, \boldsymbol{\Sigma}_l)$,

$$\boldsymbol{\mu}_l = \boldsymbol{\Sigma}_l[\alpha_2 \sum_{n=1}^N \sum_{i:c_i=l} (b_{ni}w_{ni})\boldsymbol{y}_n^{-l}], \quad \boldsymbol{\Sigma}_l = [\alpha_2 \sum_{n=1}^N \sum_{i:c_i=l} (b_{ni}w_{ni})^2 + M]^{-1}\mathbf{I}_M,$$

where $\boldsymbol{y}_n^{-l} = \boldsymbol{y}_n - \sum_{i:c_i \neq l} \mathbf{D}_i(b_{ni}w_{ni})$.

- **Update** $\{c_i\}_{i=1,T}$: $p(c_i) \sim \text{Mult}(\mathbf{p}_i)$,

$$p(c_i = l|-) \propto \begin{cases} \frac{n_l^{*-i}}{T-1+\alpha_0} \prod_{n=1}^N \exp\{-\frac{\alpha_2}{2}\|\boldsymbol{y}_n^{-i} - \mathbf{D}_l^*(b_{ni}w_{ni})\|_2^2\}, & \text{if } l \text{ is previously used,} \\ \frac{\alpha_0}{T-1+\alpha_0} \prod_{n=1}^N \exp\{-\frac{\alpha_2}{2}\|\boldsymbol{y}_n^{-i} - \mathbf{D}_{l^{new}}^*(b_{ni}w_{ni})\|_2^2\}, & \text{if } l = l^{new}, \end{cases}$$

where $n_l^{*-i} = \sum_{j:j \neq i} \delta(\mathbf{D}_j = \mathbf{D}_l^*)$, and $\boldsymbol{y}_n^{-i} = \boldsymbol{y}_n - \sum_{k:k \neq i} \mathbf{D}_k(b_{nk}w_{nk})$; $\mathbf{p}_i$ is realized by normalizing the above equation.

- **Update** $\{Z_{\boldsymbol{x}_n}\}_{n=1,N}$: for $Z_{\boldsymbol{x}_n}$, update each component $p(b_{ni}) \sim \text{Bernoulli}(v_{ni})$ for $i = 1, \ldots, K$,

$$\frac{p(b_{ni} = 1)}{p(b_{ni} = 0)} = \frac{\exp\{-\frac{\alpha_2}{2}\left[\mathbf{D}_i^T\mathbf{D}_i w_{ni}^2 - 2w_{ni}\mathbf{D}_i^T\boldsymbol{y}_n^{-i}\right]\}\pi_i K(\boldsymbol{x}_n, \boldsymbol{x}_i^*; \psi_i^*)}{1 - \pi_i K(\boldsymbol{x}_n, \boldsymbol{x}_i^*; \psi_i^*)}.$$

$v_{ni}$ is calculated by normalizing $p(b_{ni})$ with the above constraint.

- **Update** $\{\pi_i\}_{i=1,T}$:
  Introduce two sets of auxiliary variables $\{z_{ni}^{(1)}\}_{i=1,T}$ and $\{z_{ni}^{(2)}\}_{i=1,T}$ for each data $\mathbf{y}_n$. Assume $z_{ni}^{(1)} \sim \text{Bernoulli}(\pi_i)$ and $z_{ni}^{(2)} \sim \text{Bernoulli}(K(\boldsymbol{x}_n, \boldsymbol{x}_i^*; \psi_i^*))$. For each specific $n$,

  - If $b_{ni} = 1$, $z_{ni}^{(1)} = 1$ and $z_{ni}^{(2)} = 1$;

  - If $b_{ni} = 0$, $\begin{cases} p(z_{ni}^{(1)} = 0, z_{ni}^{(2)} = 0|b_{ni} = 0) = \frac{(1-\pi_i)\left(1 - K(\boldsymbol{x}_n, \boldsymbol{x}_i^*; \psi_i^*)\right)}{1 - \pi_i K(\boldsymbol{x}_n, \boldsymbol{x}_i^*; \psi_i^*)} \\ p(z_{ni}^{(1)} = 0, z_{ni}^{(2)} = 1|b_{ni} = 0) = \frac{(1-\pi_i)K(\boldsymbol{x}_n, \boldsymbol{x}_i^*; \psi_i^*)}{1 - \pi_i K(\boldsymbol{x}_n, \boldsymbol{x}_i^*; \psi_i^*)} \\ p(z_{ni}^{(1)} = 1, z_{ni}^{(2)} = 0|b_{ni} = 0) = \frac{\pi_i\left(1 - K(\boldsymbol{x}_n, \boldsymbol{x}_i^*; \psi_i^*)\right)}{1 - \pi_i K(\boldsymbol{x}_n, \boldsymbol{x}_i^*; \psi_i^*)} \end{cases}$

  From the above equations, we derive the conditional distribution for $\pi_i$,

$$\pi_i \sim \text{Beta}\left(\frac{1}{T} + \sum_n z_{ni}^{(1)}, 1 + \sum_n (1 - z_{ni}^{(1)})\right).$$

# 4 Results

## 4.1 Hyperparameter settings

For both $\alpha_1$ and $\alpha_2$ the corresponding prior was set to $\text{Gamma}(10^{-6}, 10^{-6})$; the concentration parameter $\alpha_0$ was given a prior $\text{Gamma}(1, 0.1)$. For both experiments below, the number of dictionary elements $T$ was truncated to 256, the number of unique dictionary element values was initialized to 100, and $\{\pi_i\}_{i=1,T}$ were initialized to 0.5. All $\{\psi_i^*\}_{i=1,T}$ were initialized to $10^{-5}$ and updated from a set $\{10^{-5}, 10^{-4}, 10^{-3}, 10^{-2}, 10^{-1}, 1\}$ with a uniform prior $Q$. The remaining variables were initialized randomly. No parameter tuning or optimization has been performed.

## 4.2 Music analysis

We consider the same music piece as described in [12]: "A Day in the Life" from the Beatles' album Sgt. Pepper's Lonely Hearts Club Band. The acoustic signal was sampled at 22.05 KHz and divided into 50 ms contiguous frames; 40-dimensional Mel frequency cepstral coefficients (MFCCs) were extracted from each frame, shown in Figure 1(a).

A typical goal of music analysis is to infer interrelationships within the music piece, as a function of time [12]. For the audio data, each MFCC vector $\boldsymbol{y}_n$ has an associated time index, the latter used as the covariate $\boldsymbol{x}_n$. The finite set of temporal sample points (covariates) were employed to define a library for the $\{\boldsymbol{x}_i^*\}$, and $H$ is a uniform distribution over this set. After 2000 burn-in iterations, we collected samples every five iterations. Figure 1(b) shows the frequency for the number of *unique* dictionary elements used by the data, based on the 1600 collected samples; and Figure 1(c) shows the frequency for the number of total dictionary elements used.

With the model defined in (10), the sparse vector $\boldsymbol{b}_n \circ \mathbf{w}_n$ indicates the importance of each dictionary element from $\{\mathbf{D}_i\}_{i=1,T}$ to data $\boldsymbol{y}_n$. Each of these $N$ vectors $\{\boldsymbol{b}_n \circ \mathbf{w}_n\}_{n=1,N}$ was normalized

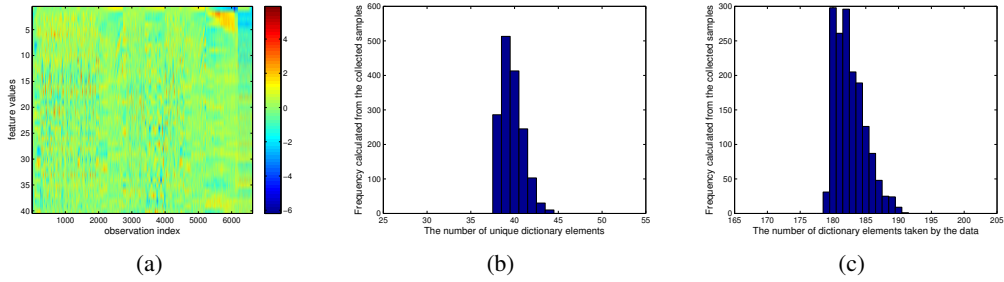

(a)                                    (b)                                    (c)

Figure 1: (a) MFCCs features used in music analysis, where the horizontal axis corresponds to time, for "A Day in the Life". Based on the Gibbs collection samples: (b) frequency on number of *unique* dictionary elements, and (c) *total* number of dictionary elements.

within each Gibbs sample, and used to compute a correlation matrix associated with the $N$ time points in the music. Finally, this matrix was averaged across the collection samples, to yield a correlation matrix relating one part of the music to all others. For a fair comparison between our methods and the model proposed in [12] (which used an HMM, and computed correlations over *windows* of time), we divided the whole piece into multiple consecutive short-time windows. Each temporal window includes 75 consecutive feature vectors, and we compute the average correlation coefficients between the features within each pair of windows. There were 88 temporal windows in total (each temporal window is de noted as a sequence in Figure 2), and the dimension of the correlation matrix is accordingly $88 \times 88$. The computed correlation matrix for the proposed KBP model is presented in Figure 2(a).

We compared KBP performance with results based on BP-FA [17] in which covariates are not employed, and with results from the dynamic clustering model in [12], in which a dynamic HMM is employed (in [12] a *dynamic* HDP, or dHDP, was used in concert with an HMM). The BP-FA results correspond to replacing the KBP with a BP. The correlation matrix computed from the BP-FA and the dHDP-HMM [12] are shown in Figures 2(b) and (c), respectively. The dHDP-HMM results yield a reasonably good segmentation of the music, but it is unable to infer subtle differences in the music over time (for example, all voices in the music are clustered together, even if they are different). Since the BP-FA does not capture as much localized information in the music (the probability of dictionary usage is the same for all temporal positions), it does not manifest as good a music segmentation as the dHDP-HMM. By contrast, the KBP-FA model yields a good music segmentation, while also capturing subtle differences in the music over time (*e.g.*, in voices). Note that the use of the DP to allow repeated use of dictionary elements as a function of time (covariates) is important here, due to the repetition of structure in the piece. One may listen to the music and observe the segmentation at `http://www.youtube.com/watch?v=35YhHEbIlEI`.

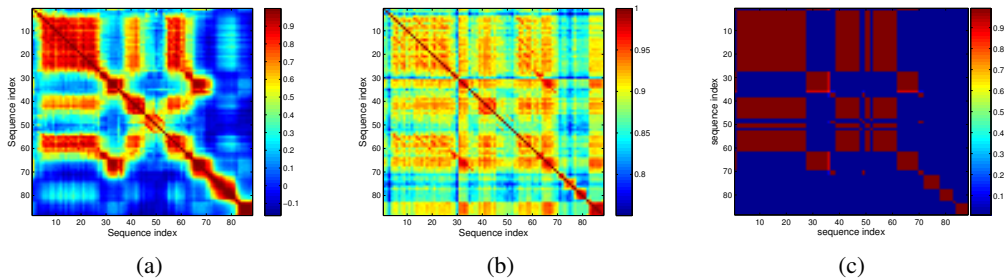

(a)                                    (b)                                    (c)

Figure 2: Inference of relationships in music as a function of time, as computed via a correlation of the dictionary-usage weights, for (a) and (b), and based upon state usage in an HMM, for (c). Results are shown for "A Day in the Life." The results in (c) are from [12], as a courtesy from the authors of that paper. (a) KBP-FA, (b) BP-FA, (c) dHDP-HMM .

### 4.3 Image interpolation and denoising

We consider image interpolation and denoising as two additional potential applications. In both of these examples each image is divided into $N$ $8 \times 8$ overlapping patches, and each patch is stacked into a vector of length $M = 64$, constituting observation $\boldsymbol{y}_n \in \mathbb{R}^M$. The covariate $\boldsymbol{x}_n$ represents the

patch coordinates in the 2-D space. The probability measure $H$ corresponds to a uniform distribution over the centers of all $8 \times 8$ patches. The images were recovered based on the average of the collection samples, and each pixel was averaged across all overlapping patches in which it resided. For the image-processing examples, 5000 Gibbs samples were run, with the first 2000 discarded as burn-in.

For image interpolation, we only observe a fraction of the image pixels, sampled uniformly at random. The model infers the underlying dictionary $\mathbf{D}$ in the presence of this missing data, as well as the weights on the dictionary elements required for representing the observed components of $\{\boldsymbol{y}_n\}$; using the inferred dictionary and associated weights, one may readily impute the missing pixel values. In Table 1 we present average PSNR values on the recovered pixel values, as a function of the fraction of pixels that are observed (20% in Table 1 means that 80% of the pixels are missing uniformly at random). Comparisons are made between a model based on BP and one based on the proposed KBP; the latter generally performs better, particularly when a large fraction of the pixels are missing. The proposed algorithm yields results that are comparable to those in [18], which also employed covariates within the BP construction. However, the proposed KBP construction has the significant computational advantages of only requiring kernels centered at the locations of the dictionary-dependent covariates $\{\boldsymbol{x}_i^*\}$, while the model in [18] has a kernel for each of the image patches, and therefore it scales unfavorably for large images.

Table 1: Comparison of BP and KBP for interpolating images with pixels missing uniformly at random, using standard image-processing images. The top and bottom rows of each cell show results of BP and KBP, respectively. Results are shown when 20%, 30% and 50% of the pixels are observed, selected uniformly at random.

| RATIO | C.MAN | HOUSE | PEPPERS | LENA | BARBARA | BOATS | F.PRINT | MAN | COUPLE | HILL |
|---|---|---|---|---|---|---|---|---|---|---|
| 20% | 23.75 | 29.75 | 25.56 | 30.97 | 26.84 | 27.84 | 26.49 | 28.29 | 27.76 | 29.38 |
| | **24.02** | **30.89** | **26.29** | **31.38** | **28.93** | **28.11** | **26.89** | **28.37** | **28.03** | **29.67** |
| 30% | 25.59 | 33.09 | 28.64 | 33.30 | 30.13 | 30.20 | 29.23 | 29.89 | 29.97 | 31.19 |
| | **25.75** | **34.02** | **29.29** | **33.33** | **31.46** | **30.24** | **29.37** | **30.12** | **30.33** | **31.25** |
| 50% | 28.66 | 38.26 | 32.53 | **36.79** | 35.95 | 33.05 | **33.50** | **33.19** | **33.61** | **34.19** |
| | **28.78** | **38.35** | **32.69** | 35.89 | **36.03** | **33.18** | 32.18 | 32.35 | 32.35 | 32.60 |

In the image-denoising example in Figure 3 the images were corrupted with both white Gaussian noise (WGN) and sparse spiky noise, as considered in [18]. The sparse spiky noise exists in particular pixels, selected uniformly at random, with amplitude distributed uniformly between $-255$ and $255$. For the pepper image, $15\%$ of the pixels were corrupted by spiky noise, and the standard deviation of the WGN was 15; for the house image, $10\%$ of the pixels were corrupted by spiky noise and the standard deviation of WGN was 10. We compared with different methods on both two images: the augmented KBP-FA model (KBP-FA+) in Sec. 3.2, the BP-FA model augmented with a term for spiky noise (BP-FA+) and the original BP-FA model. The model proposed with KBP showed the best denoising result for both visual and quantitative evaluations. Again, these results are comparable to those in [18], with the significant computational advantage discussed above. Note that here the imposition of covariates and the KBP yields marked improvements in this application, relative to BP-FA alone.

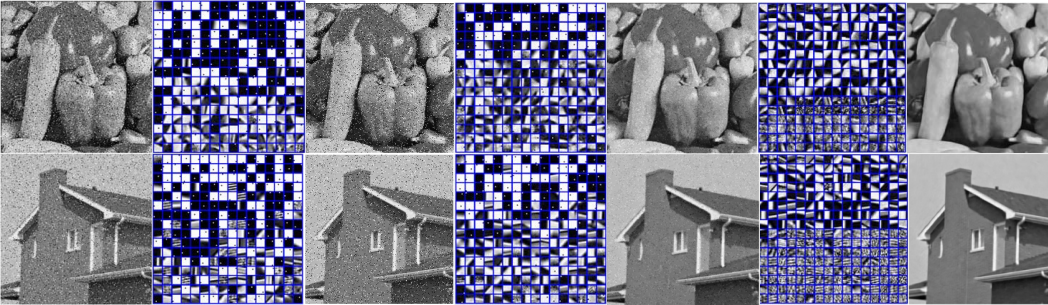

Figure 3: Denoising Result: the first column shows the noisy images (PSNR is 15.56 dB for Peppers and 17.54 dB for House); the second and third column shows the results inferred from the BP-FA model (PSNR is 16.31 dB for Peppers and 17.95 dB for House), with the dictionary elements shown in column two and the reconstruction in column three; the fourth and fifth columns show results from BP-FA+ (PSNR is 23.06 dB for Peppers and 26.71 dB for House); the sixth and seventh column shows the results of the KBP-FA+ (PSNR is 27.37 dB for Peppers and 34.89 dB for House). In each case the dictionaries are ordered based on their frequency of usage, starting from top-left.

# 5 Summary

A new Lévy process, the kernel beta process, has been developed for the problem of nonparametric Bayesian feature learning, with example results presented for music analysis, image denoising, and image interpolation. In addition to presenting theoretical properties of the model, state-of-the-art results are realized on these learning tasks. The inference is performed via a Gibbs sampler, with analytic update equations. Concerning computational costs, for the music-analysis problem, for example, the BP model required around 1 second per Gibbs iteration, with KBP requiring about 3 seconds, with results run on a PC with 2.4GHz CPU, in non-optimized Matlab™.

## Acknowledgment

The research reported here was supported by AFOSR, ARO, DARPA, DOE, NGA and ONR.

## References

[1] D. Applebaum. *Levy Processes and Stochastic Calculus*. Cambridge University Press, 2009.

[2] D. B. Dunson and J.-H. Park. Kernel stick-breaking processes. *Biometrika*, 95:307–323, 2008.

[3] T. Ferguson. A Bayesian analysis of some nonparametric problems. *The Annals of Statistics*, 1973.

[4] T. L. Griffiths and Z. Ghahramani. Infinite latent feature models and the Indian buffet process. In *NIPS*, 2005.

[5] N. L. Hjort. Nonparametric Bayes estimators based on beta processes in models for life history data. *Annals of Statistics*, 1990.

[6] J.F.C. Kingman. *Poisson Processes*. Oxford Press, 2002.

[7] D. Knowles and Z. Ghahramani. Infinite sparse factor analysis and infinite independent components analysis. In *Independent Component Analysis and Signal Separation*, 2007.

[8] S. N. MacEachern. Dependent Nonparametric Processes. In *In Proceedings of the Section on Bayesian Statistical Science*, 1999.

[9] K. Miller, T. Griffiths, and M. I. Jordan. The phylogenetic Indian buffet process: A non-exchangeable nonparametric prior for latent features. In *UAI*, 2008.

[10] C.E. Rasmussen and C. Williams. *Gaussian Processes for Machine Learning*. MIT Press, 2006.

[11] L. Ren, L. Du, L. Carin, and D. B. Dunson. Logistic stick-breaking process. *J. Machine Learning Research*, 2011.

[12] L. Ren, D. Dunson, S. Lindroth, and L. Carin. Dynamic nonparametric bayesian models for analysis of music. *Journal of The American Statistical Association*, 105:458–472, 2010.

[13] A. Rodriguez and D. B. Dunson. Nonparametric bayesian models through probit stickbreaking processes. *Univ. California Santa Cruz Technical Report*, 2009.

[14] J. Sethuraman. A constructive definition of dirichlet priors. 1994.

[15] R. Thibaux and M. I. Jordan. Hierarchical beta processes and the Indian buffet process. In *AISTATS*, 2007.

[16] S. Williamson, P. Orbanz, and Z. Ghahramani. Dependent Indian buffet processes. In *AISTATS*, 2010.

[17] M. Zhou, H. Chen, J. Paisley, L. Ren, G. Sapiro, and L. Carin. Non-parametric Bayesian dictionary learning for sparse image representations. In *NIPS*, 2009.

[18] M. Zhou, H. Yang, G. Sapiro, D. Dunson, and L. Carin. Dependent hierarchical beta process for image interpolation and denoising. In *AISTATS*, 2011.

